# Dopamine modulation in a basal ganglio-cortical network implements saliency-based gating of working memory

**Aaron J. Gruber**[1,2]**, Peter Dayan**[3]**, Boris S. Gutkin**[3]**, and Sara A. Solla**[2,4]

Biomedical Engineering[1], Physiology[2], and Physics and Astronomy[4],
Northwestern University, Chicago, IL, USA.
Gatsby Computational Neuroscience Unit[3],
University College London, London, UK.
{*a-gruber1,solla* }*@northwestern.edu*, {*dayan,boris*}*@gatsby.ucl.ac.uk*

## Abstract

Dopamine exerts two classes of effect on the sustained neural activity in prefrontal cortex that underlies working memory. Direct release in the cortex increases the contrast of prefrontal neurons, enhancing the robustness of storage. Release of dopamine in the striatum is associated with salient stimuli and makes medium spiny neurons bistable; this modulation of the output of spiny neurons affects prefrontal cortex so as to indirectly gate access to working memory and additionally damp sensitivity to noise. Existing models have treated dopamine in one or other structure, or have addressed basal ganglia gating of working memory exclusive of dopamine effects. In this paper we combine these mechanisms and explore their joint effect. We model a memory-guided saccade task to illustrate how dopamine's actions lead to working memory that is selective for salient input and has increased robustness to distraction.

## 1 Introduction

Ample evidence indicates that the maintenance of information in working memory (WM) is mediated by persistent neural activity in the prefrontal cortex (PFC) [9, 10]. Critical for such memories is to control how salient external information is gated into storage, and to limit the effects of noise in the neural substrate of the memory itself. Experimental [15, 18] and theoretical [2, 13, 4, 17] studies implicate dopaminergic neuromodulation of PFC in information gating and noise control. In addition, there is credible speculation [7] that input to the PFC from the basal ganglia (BG) should also exert gating effects. Since the striatum is also a major target of dopamine innervation, the nature of the interaction between these various control structures and mechanisms in manipulating WM is important.

A wealth of mathematical and computational models bear on these questions. A recent cellular-level model, which includes many known effects of dopamine (DA) on ionic conductances, indicates that modulation of pyramidal neurons causes the pattern of network activity at a fixed point attractor to become more robust both to noise and to input-driven

switching of attractor states [6]. This result is consistent with reported effects of DA in more abstract, spiking-based models [2] of WM, and provides a cellular substrate for network models that account for gating effects of DA in cognitive WM tasks [1]. Other network models [7] of cognitive tasks have concentrated on the input from the BG, arguing that it has a disinhibitory effect (as in models of motor output) that controls bistability in cortical neurons and thereby gates external input to WM. This approach emphasizes the role of dopamine in providing a training signal to the BG, in contrast to the modulatory effects of DA discussed here, which are important for on-line neural processing. Finally, dopaminergic neuromodulation in the striatum has itself been recently captured in a biophysically-grounded model [11], which describes how medium spiny neurons (MSNs) become bistable in elevated dopamine. As the output of a major subset of MSNs ultimately reaches PFC after further processing through other nuclei, this bistability can have potentially strong effects on WM.

In this paper, we combine these various influences on working memory activity in the PFC. We model a memory-guided saccade task [8] in which subjects must fixate on a centrally located fixation spot while a visual target is flashed at a peripheral location. After a delay period of up to a few seconds, subjects must saccade to the remembered target location. Numerous experimental studies of the task show that memory is maintained through striatal and sustained prefrontal neuronal activity; this persistent activity is consistent with attractor dynamics. Robustness to noise is of particular importance in the WM storage of continuous scalar quantities such as the angular location of a saccade target, since internal noise in the attractor network can easily lead to drift in the activity encoding the memory. In successive sections of this paper, we consider the effect of DA on resistance to attractor switching in the isolated cortical network; the effect of MSN activity on gating and noise; and the effect of dopamine induced bistability in MSNs on WM activity associated with salient stimuli. We demonstrate that DA exerts complementary direct and indirect effects, which result in superior performance in memory-guided tasks.

## 2   Model description

The components of the network model used to simulate the WM activity during a memory-guided saccade task are shown in Fig 1. The input module consists of a ring of 120 units that project both to the PFC and the BG modules. Input units are assigned firing rates $r_j^T$ to represent the sensory cortical response to visual targets. Bumps of activity centered at different locations along the ring encode for the position of different targets around the circle, as characterized by an angle in the $[0, 2\pi)$ interval.

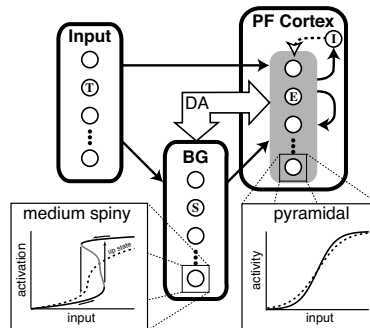

Figure 1: The network model consists of three modules: cortical input, basal ganglia (BG), and prefrontal cortex (PFC). Insets show the response functions of spiny (BG) and pyramidal (PFC) neurons for both low (dotted curves) and high (solid curves) dopamine.

The BG module consists of 24 medium spiny neurons (MSNs). Connections from the input units consist of Gaussian receptive fields that assign to each MSN a preferred direction; these preferred directions are monotonically and uniformly distributed. The dynamics of individual MSNs follow from a biophysically-grounded single compartment model [11]

$$-C\dot{V}^S = \gamma \left(I_{\text{IRK}} + I_{\text{LCa}}\right) + I_{\text{ORK}} + I_{\text{L}} + I_T, \tag{1}$$

which incorporates three crucial ionic currents: an inward rectifying $K^+$ current ($I_{\text{IRK}}$), an outward rectifying $K^+$ current ($I_{\text{ORK}}$), and an $L$-type $Ca^{2+}$ current ($I_{\text{LCa}}$). The characterization of these currents is based on available biophysical data on MSNs. The factor $\gamma$ represents an increase in the magnitude of the $I_{\text{IRK}}$ and $I_{\text{LCa}}$ currents due to the activation of D1 dopamine receptors. This DA induced current enhancement renders the response function of MSNs bistable for $\gamma \gtrsim 1.2$ (see Fig 1 for $\gamma = 1.4$). The synaptic input $I_T$ is an ohmic term with conductance given by the weighted summed activity of the corresponding input unit; input to the $j$-th MSN is thus given by $I_{Tj} = \sum_i W_{ji}^{ST} r_i^T V_j^S$, where $W_{ji}^{ST}$ is the strength of the connection from the $i$-th input neuron to the $j$-th spiny neuron. The firing rate of MSNs is a logistic function of their membrane potential: $r_j^S = L(V_j^S)$. The MSNs provide excitatory inputs to the PFC; in the model, this monosynaptic projection represents the direct pathway through the globus pallidus/substantia nigra and thalamus.

The PFC module implements a line attractor capable of sustaining a bump of activity that encodes for the value of an angular variable in $[0, 2\pi)$. 'Bump' networks like this have been used [3, 5] to model head direction and visual stimulus location characterized by a single angular variable. The module consists of 120 excitatory units; each unit is assigned a preferred direction, uniformly covering the $[0, 2\pi)$ interval. Lateral connections between excitatory units are a Gaussian function of the angular difference between the corresponding preferred directions. A single inhibitory unit provides uniform global inhibition; the activity of the inhibitory unit is controlled by the total activity of the excitatory population. This type of connectivity guarantees that a localized bump of activity, once established, will persist beyond the disappearance of the external input that originated it (see Fig 2). One of the purposes of this paper is to investigate whether this persistent activity bump is robust to noise in the line attractor network.

The excitatory units follow the stochastic differential equation

$$\tau^E \dot{V}_j^E = -V_j^E + \sum_i W_{ji}^{ES} r_i^S + \sum_{i \neq j} W_{ji}^{EE} r_i^E - r^I + r_j^T + \sigma_e \eta. \qquad (2)$$

The first sum in Eq 2 represents inputs from the BG; the connections $W_{ji}^{ES}$ consist of Gaussian receptive fields centered to align with the preferred direction of the corresponding excitatory unit. The second sum represents inputs from other excitatory PFC units; note that self-connections are excluded. The following two terms represent input from the inhibitory PFC unit ($r^I$) and information about the visual target provided by the input module ($r_j^T$). Crucially, the last term provides a stochastic input that models fluctuations in the activities that contribute to the total input to the excitatory units. The random variable $\eta$ is drawn from a Gaussian distribution with zero mean and unit variance. The noise amplitude $\sigma_e$ scales like $(dt)^{-1/2}$, where $dt$ is the integration time step. The firing rate of the PFC excitatory units is a logistic function $r_j^E = L(V_j^E)$; as shown in Fig 1, the steepness of this response function is controlled by DA. The dynamics of the inhibitory unit follows from $\tau^I \dot{V}^I = \sum_i r_i^E$, where the sum represents the total activity of the excitatory population. The firing rate $r^I$ of the inhibitory unit is a linear threshold function of $V^I$. Dopaminergic modulation of the PFC network is implemented through an increase in the steepness of the response function of the excitatory cortical units. Gain control of this form has been adopted in a previous, more abstract, network theory of WM [17], and is generally consistent with biophysically-grounded models [6, 2].

To investigate the properties of the network model represented in Fig 1, the system of equations summarized above is integrated numerically using a 5th order Runge-Kutta method with variable time step that ensures an error tolerance below 5 $\mu$V/ms.

## 3 Results

*3.1 Dopamine effects on the cortex: increased memory robustness*

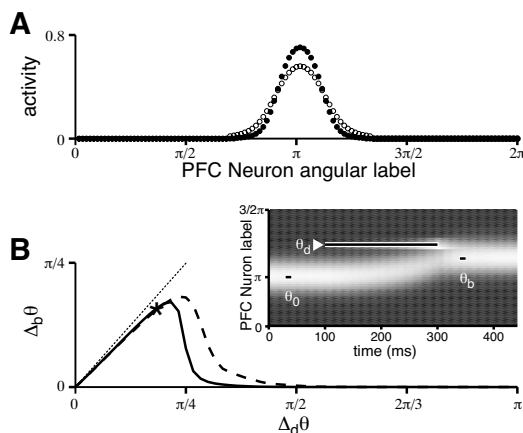

Figure 2: (A) Activity profile of the bump state in low DA (open dots) and high DA (full dots). (B) Robustness characteristics of bump activity in low DA (dashed curve) and high DA (solid curve). For reference, the thin dotted line indicates the identity $\Delta_b\theta = \Delta_d\theta$. The activity profile shown as a function of time in the inset (grey scale, white as most active) illustrates the displacement of the bump from its initial location at $\theta_0$ to a final location at $\theta_b$ due to a distractor input at $\theta_d$. This case corresponds to the asterisk on the curves in B.

We first investigate the properties of the cortical network isolated from the input and basal ganglia components. The connectivity among cortical units is set so there are two stable states of activity for the PFC network: either all excitatory units have very low activity level, or a subset of them participates in a localized bump of elevated activity (Fig 2A, open dots). The bump can be translated to any position along the ring of cortical units, thus providing a way to encode a continuous variable, such as the angular position of a stimulus within a circle. The encoded angle corresponds to the location of the bump peak, and it can be read out by computing the population vector. The effect of DA on the PFC module, modeled here as an increase in the gain of the response function of the excitatory units, results in a narrower bump with a higher peak (Fig 2A, full dots).

We measure the robustness of the location of the bump state against perturbative distractor inputs by applying a brief distractor at an angular distance $\Delta_d\theta$ from the current location of the bump and assessing the resulting angular displacement $\Delta_b\theta$ in the location of the bump 40 ms after the offset of the distractor. The procedure is illustrated in the inset of Fig 2B, which shows that a distractor current injection centered at a location $\theta_d$ causes a drift in bump location from its initial position $\theta_0$ to a final position $\theta_b$, closer to the angular location of the distractor. If $\theta_d$ is close to $\theta_0$, the distractor is capable of moving the bump completely to the injection location, and $\Delta_b\theta$ is almost equal to $\Delta_d\theta$. As shown in Fig 2B, the plot of $\Delta_b\theta$ versus $\Delta_d\theta$ remains close to the identity line for small $\Delta_d\theta$. However, as $\Delta_d\theta$ increases the distractor becomes less and less effective, until the displacement $\Delta_b\theta$ of the bump decreases abruptly and becomes negligible.

The generic features of bump stability shown in Fig 2B apply to both low DA (dashed curve) and high DA (solid curve) conditions. The difference between these two curves reveals that the dopamine induced increase in the gain of PFC units *decreases* the sensitivity of the bump to distractors, resulting in a consistently smaller bump displacement. The actual location of these two curves can be altered by varying the intensity and/or the duration of the distractor input, but their features and relative order remain invariant. This numerical experiment demonstrates that DA increases the robustness of the encoded memory, consistent with other PFC models of DA effects on WM [2, 6].

### 3.2 Basal ganglia effects on the cortex: increased memory robustness and input gating

Next, we investigate the effects of BG input (both tonic and phasic) on the stability of PFC bump activity in the absence of DA modulation. Tonic input from a single MSN, whose preferred direction coincides with the angular location of the bump, anchors the bump at that location and increases memory robustness against both noise induced diffusion (Figs

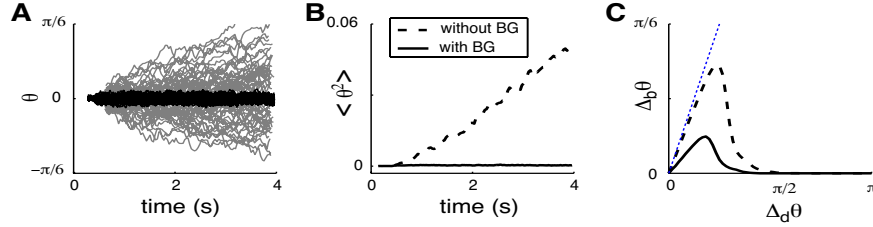

Figure 3: Diffusion of the bump location due to noise in low DA (grey traces in A; dashed curve in B) is greatly reduced by input from a single BG unit with the same preferred angular location (dark traces in A; solid curve in B). The robustness to distractor driven drift is also increased by BG input (C).

3A and 3B) and distractors (Fig 3C). Such localized tonic input to the PFC effectively breaks the symmetry of the line attractor, yielding a single fixed point for the cortical active state: a bump centered at the location of maximal BG input. This transition from a continuous line attractor to a fixed point attractor reduces the maximal deviation of the bump by a distractor.

Active MSNs provide control over the encoded memory not only by enhancing robustness, as shown above for the case of tonic input to the PFC, but also by providing phasic input that can assist a relevant visual stimulus in switching the location of the PFC activity bump. We show in Fig 4 (top plots) the location of the activity bump $\theta_b$ as a function of time in response to two stimuli at different locations $\theta_s$. The nature of the PFC response to the second stimulus depends dramatically on whether it elicits activity in the MSNs. The initial stimulus activates a tight group of MSNs which encode for its angular position. It also causes activation of a group of PFC neurons whose population vector encodes for the same angular position. When the input disappears, the MSNs become inactive and the cortical layer relaxes to a characteristic bump state centered at the angular position of the stimulus. A second stimulus (distractor) that fails to activate BG units (Fig 4A) has only a minimal effect on the bump location. However, if the stimulus *does* activate the BG units (Fig 4B), then it causes a switch in bump location. In this case, the PFC memory is updated to encode for the location of the most recent stimulus. Thus a direct stimulus input to the PFC that by itself is not sufficient to switch attractor states can trigger a switch, provided it activates the BG, whose activity yields additional input to the PFC. Transient activation of MSNs thus effectively gates access to working memory.

### 3.3 Dopamine effects on the basal ganglia: saliency-based gating

Ample evidence indicates that DA, the release of which is associated with the presentation of conditioned stimuli [16], modulates the activity of MSNs. Our previous computational model of MSNs [11] studied the apparently paradoxical effects of DA modulation, manifested in both suppression and enhancement of MSN activity in a complex reward-based saccade task [12]. We showed that DA can induce bistability in the response functions of MSNs, with important consequences. In high DA, the effective threshold for reaching the active 'up' state is increased; the activity of units that do not exceed threshold is suppressed into a quiescent 'down' state, while units that reach the up state exhibit a higher firing rate which is extended in duration due to effects of hysteresis.

We now demonstrate that the dual enhancing/suppressing nature of DA modulation of MSNs activity significantly affects the network's response to stimuli. We show in Fig 5 (top plot) the location of the activity bump $\theta_b$ as a function of time in response to four stimuli at two different locations: $\theta_A, \theta_B, \theta_A^*, \theta_B$. Crucially, in this sequence, only $\theta_A^*$ is a conditioned stimulus that triggers DA release.

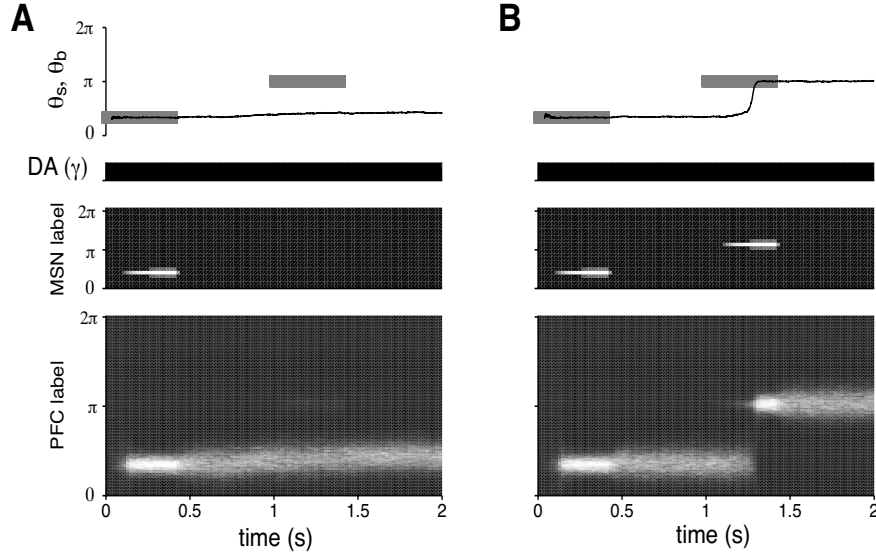

Figure 4: Top plot shows the location $\theta_b$ of the encoded memory as determined from the population vector of the excitatory cortical units (thin black curve) and the location $\theta_s$ of stimuli as encoded by a Gaussian bump of activity in the input units (grey bars) as a function of time. The middle and bottom panels show the activity of the BG and the PFC modules, respectively. Dopamine level remains low.

The first two stimuli activate appropriate MSNs, and are therefore gated into WM. The presentation of $\theta_A^*$ activates the same set of MSNs as $\theta_A$, but the DA-modulated MSNs now become bistable: high activity is enhanced while intermediate activity is suppressed. Only the central MSN remains active with an enhanced amplitude; the two lateral MSNs that were transiently activated by $\theta_A$ in low DA are now suppressed. The activity of the central MSN suffices to gate the location of the new stimulus into WM; the location of the PFC activity bump switches accordingly. Interestingly, this switch from B to A occurs more slowly than the preceding switch from A to B. This effect is also attributable to DA: its release affects the response function of excitatory PFC units, making them less likely to react to a subsequent stimulus and thus enhancing the stability of the bump at the $\theta_B$ angular position. Once the bump has switched to the angular location $\theta_A^*$ to encode for the conditioned stimulus, the subsequent presentation of $\theta_B$ does not activate MSNs since they are hysteretically locked in the inactive down state. The pattern of activity in the BG continues to encode for $\theta_A$ for as long as the DA level remains elevated, and the PFC activity bump continues to encode for $\theta_A^*$.

In sum, DA induced bistability of MSNs, associated with an expectation of reward, imparts salience selectivity to the gating function of the BG. By locking the activation of MSNs associated with salient input, the BG input prevents a switch in PFC bump activity and preserves the conditioned stimulus in WM. The robustness of the WM activity is enhanced by a combined effect of DA through both increasing the gain of PFC neurons and sustaining MSN input during the delay period (see Fig 5, bottom plot).

## 4 Discussion

We have built a working memory model which links dopaminergic neuromodulation in the prefrontal cortex, bistability-inducing dopaminergic neuromodulation of striatal spiny

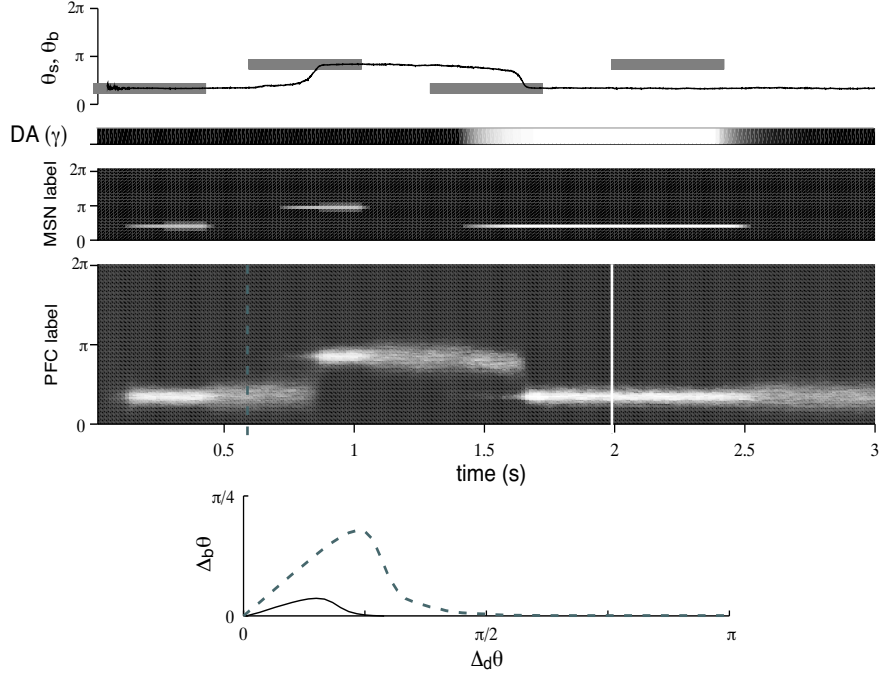

Figure 5: Top plot shows the location $\theta_b$ of the encoded memory as determined from the population vector of the excitatory cortical units (thin black curve) and the location $\theta_s$ of stimuli as encoded by a Gaussian bump of activity in the input units (grey bars) as a function of time. The second and third panels bottom plots show the activity of the BG and the PFC modules, respectively. Dopamine level increases in response to the conditioned stimulus. The bottom plot displays increased robustness of WM for conditioned (solid curve) as compared to unconditioned (dashed curve) stimuli.

neurons, and the effects of basal ganglia output on cortical persistence. The resulting interactions provide a sophisticated control mechanism over the read-in to working memory and the elimination of noise. We demonstrated the quality of the system in a model of a standard memory-guided saccade task.

There are two central issues for models of working memory: robustness to *external* noise, such as explicit lures presented during the memory delay period, and robustness to *internal* noise, coming from unwarranted corruption of the neural substrate of persistent activity. Our model, along with various others, addresses these issues at a cortical level via two basic mechanisms: DA modulation, which changes the excitability of neurons in a particular way (units that are inactive are less excitable by input, while units that are active can become more active), and targeted input from the BG. However, models differ as to the nature and provenance of the BG input, and also its effects on the PFC. Ours is the first to consider the combined, complementary, effects of DA in the PFC and the BG.

The requirements for a gating signal are that it be activated at the same time as the stimuli that are to be stored, and that it is a (possibly exclusive) means by which a WM state is established. Following the experimental evidence that perturbing DA leads to disruption of WM [18], a set of theories suggested that a phasic DA signal (as associated, for instance, with reward predicting conditioned stimuli [16]) acts as the gate in the cortex [4]. In various models [17, 2, 6], and also in ours, phasic DA is able to act as a gate through its contrast-enhancing effect on cortical activity. However, as discussed at length in Frank

*et al* [7] (whose model does not incorporate the effect at all), this is unlikely to be the sole gating mechanism, since various stimuli that would not lead to the release of phasic DA still require storage in WM. In our model, even in low DA, the BG gates information by controlling the switching of the attractor state in response to inputs. Frank *et al* [7] point out the various advantages of this type of gating, largely associated with the opportunities for precise temporal and spatial gating specificity, based on information about the task context.

Our BG gating mechanism simply involves additional targeted excitatory input to the cortex from the (currently over-simplified) output of striatal spiny neurons, coupled with a detailed account [11] of DA induced bistability in MSNs. This allows us to couple gating to motivationally salient stimuli that induce the release of DA. Since DA controls plasticity in cortico-striatal synapses [14], there is an available mechanism for learning the appropriate gating of salient stimuli, as well as motivationally neutral contextual stimuli that do not trigger DA release but are important to store.

Robustness against noise that is internal to the WM is of particular importance for line or surface attractor memories, since they have one or more global directions of null stability and therefore exhibit propensity to diffuse. Rather than rely on bistability in cortical neurons [3], our model relies on input from the striatum to reduce drift. This mechanism is available in both high and low DA conditions. This additional input turns the line attractor into a point attractor at the given location, and thereby adds stability while it persists. The DA induced bistability of MSNs, for which there is now experimental evidence, enhances this stabilization effect.

We have focused on the mechanisms by which DA and the BG can influence WM. An important direction for future work is to relate this material to our growing understanding of the provenance of the DA signal in terms of reward prediction errors and motivationally salient cues.

## References

[1] Braver TS, Cohen JD (1999) *Prog. Brain Res.* 121:327-349.

[2] Brunel N, Wang XJ (2001) *J. Comp. Neurosci.* 11:63-85.

[3] Camperi M, Wang XJ (1998) *J. Comp. Neurosci.* 5:383-405.

[4] Cohen JD, Braver TS, Brown JW (2002) *Curr. Opin. Neurobiol.* 12:223-229.

[5] Compte A, Brunel N, Goldman-Rakic P, Wang XJ (2000) *Cereb. Cortex* 10:910-923.

[6] Durstewitz D, Seamans J, Sejnowski T (2000) *J. Neurophys.* 83:1733-1750.

[7] Frank M, Loughry B, O'Reilly RC (2001) *Cog., Affective, & Behav. Neurosci.* 1(2):137-160.

[8] Funahashi S, Bruce CJ, Goldman-Rakic PS (1989) *J. Neurophys.* 255:556-559.

[9] Fuster J (1995) Memory in the Cerebral Cortex *MIT Press*.

[10] Goldman-Rakic PS (1995) *Neuron* 14:477-85.

[11] Gruber AJ, Solla SA, Houk JC (2003). *NIPS* 15.

[12] Kawagoe R, Takikawa Y, Hikosaka O (1998) *Nat. Neurosci.* 1:411-416.

[13] O'Reilly RC, Noelle DC, Braver TS, Cohen JD (2002) *Cerebral Cortex* 12:246-257.

[14] Reynolds JN, Wickens JR (2000) *Neurosci.* 99:199-203.

[15] Sawaguchi T, Goldman-Rakic PS (1991) *Science* 251:947-950.

[16] Schultz W, Apicella P, Ljungberg T (1993) *J. Neurosci.* 13:900-913.

[17] Servan-Schreiber D, Printz H, Cohen J (1990) *Science* 249:892-895.

[18] Williams GV, Goldman-Rakic PS (1995) *Nature* 376:572-575.
